# Optimal Filtering in the Salamander Retina

Fred Rieke[a,c], W. Geoffrey Owen[b] and William Bialek[a,b,c]

Departments of Physics[a] and Molecular and Cell Biology[b]
University of California
Berkeley, California 94720
and
NEC Research Institute[c]
4 Independence Way
Princeton, New Jersey 08540

## Abstract

The dark-adapted visual system can count photons with a reliability limited by thermal noise in the rod photoreceptors — the processing circuitry between the rod cells and the brain is essentially noiseless and in fact may be close to optimal. Here we design an optimal signal processor which estimates the time-varying light intensity at the retina based on the rod signals. We show that the first stage of optimal signal processing involves passing the rod cell output through a linear filter with characteristics determined entirely by the rod signal and noise spectra. This filter is very general; in fact it is the first stage in any visual signal processing task at low photon flux. We identify the output of this first-stage filter with the intracellular voltage response of the bipolar cell, the first anatomical stage in retinal signal processing. From recent data on tiger salamander photoreceptors we extract the relevant spectra and make parameter-free, quantitative predictions of the bipolar cell response to a dim, diffuse flash. Agreement with experiment is essentially perfect. As far as we know this is the first successful predictive theory for neural dynamics.

## 1  Introduction

A number of biological sensory cells perform at a level which can be called optimal — their performance approaches limits set by the laws of physics [1]. In some cases

the behavioral performance of an organism, not just the performance of the sensory cells, also approaches fundamental limits. Such performance indicates that neural computation can reach a level of precision where the reliability of the computed output is limited by noise in the sensory input rather than by inefficiencies in the processing algorithm or noise in the processing hardware [2]. These observations suggest that we study algorithms for optimal signal processing. If we can make the notion of optimal processing precise we will have the elements of a predictive (and hence unequivocally testable) theory for what the nervous system *should* compute. This is in contrast to traditional modeling approaches which involve adjustment of free parameters to fit experimental data.

To further develop these ideas we consider the vertebrate retina. Since the classic experiments of Hecht, Shlaer and Pirenne we have known that the dark-adapted visual system can count small numbers of photons [3]. Recent experiments confirm Barlow's suggestion [4,5] that the reliability of behavioral decision making reaches limits imposed by dark noise in the photoreceptors due to thermal isomerization of the photopigment [6]. If dark-adapted visual performance is limited by thermal noise in the sensory cells then the subsequent layers of signal processing circuitry must be extremely reliable. Rather than trying to determine precise limits to reliability, we follow the approach introduced in [7] and use the notion of "optimal computation" to design the optimal processor of visual stimuli. These theoretical arguments result in parameter-free predictions for the dynamics of signal transfer from the rod photoreceptor to the bipolar cell, the first stage in visual signal processing. We compare these predictions directly with measurements on the intact retina of the tiger salamander *Ambystoma tigrinum* [8,9].

## 2    Design of the optimal processor

All of an organism's knowledge of the visual world derives from the currents $I_n(t)$ flowing in the photoreceptor cells (labeled n). Visual signal processing consists of estimating various aspects of the visual scene from observation of these currents. Furthermore, to be of use to the organism these estimates must be carried out in real time. The general problem then is to formulate an optimal strategy for estimating some functional $G[R(\mathbf{r}, t)]$ of the time and position dependent photon arrival rate $R(\mathbf{r}, t)$ from real time observation of the currents $I_n(t)$.

We can make considerable analytic progress towards solving this general problem using probabilistic methods [7,2]. Start by writing an expression for the probability of the functional $G[R(\mathbf{r}, t)]$ conditional on the currents $I_n(t)$, $P\{G[R(\mathbf{r}, t)]|I_n(t)\}$. Expanding for low signal-to-noise ratio (SNR) we find that the first term in the expansion of $P\{G|I\}$ depends only on a filtered version of the rod currents,

$$P\{G[R(\mathbf{r}, t)]|I_n(t)\} = \Im_G[F * I_n] + \text{higher order corrections}, \qquad (1)$$

where $*$ denotes convolution; the filter $F$ depends only on the signal and noise characteristics of the photoreceptors, as described below. Thus the estimation task divides naturally into two stages — a universal "pre-processing" stage and a task-dependent stage. The universal stage is independent both of the stimulus $R(\mathbf{r}, t)$ and of the particular functional $G[R]$ we wish to estimate. Intuitively this separation makes sense; in conventional signal processing systems detector outputs are first

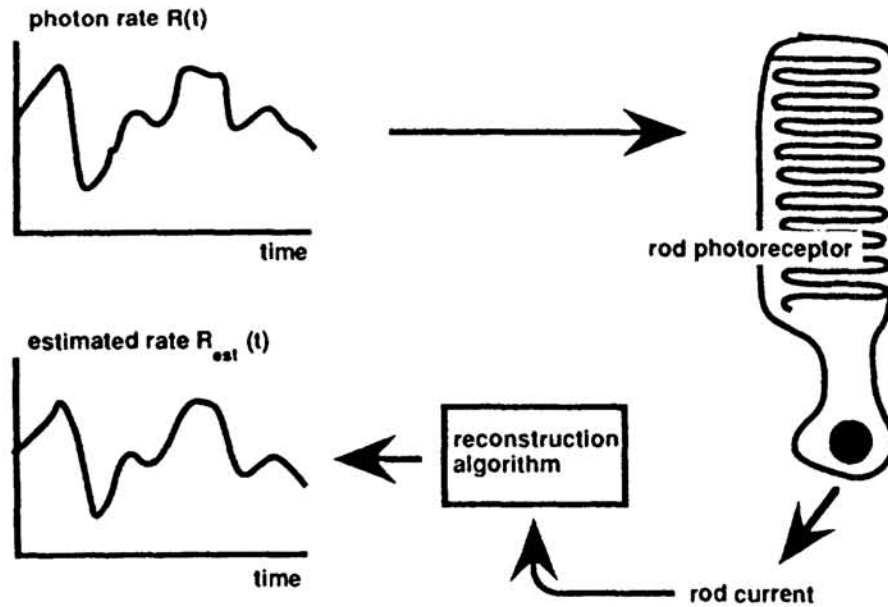

Figure 1: Schematic view of photon arrival rate estimation problem.

processed by a filter whose shape is motivated by general SNR considerations. Thus the view of retinal signal processing which emerges from this calculation is a preprocessing or "cleaning up" stage followed by more specialized processing stages. We emphasize that this separation is a mathematical fact, not a model we have imposed.

To fill in some of the details of the calculation we turn to the simplest example of the estimation tasks discussed above — estimation of the photon arrival rate itself (Fig. 1): Photons from a light source are incident on a small patch of retina at a time-varying rate $R(t)$, resulting in a current $I(t)$ in a particular rod cell. The theoretical problem is to determine the optimal strategy for estimating $R(t)$ based on the currents $I(t)$ in a small collection of rod cells. With an appropriate definition of "optimal" we can pose the estimation problem mathematically and look for analytic or numerical solutions. One approach is the conditional probability calculation discussed above [7]. Alternatively we can solve this problem using functional methods. Here we outline the functional calculation.

Start by writing the estimated rate as a filtered version of the rod currents:

$$R_{est}(t) = \int d\tau F_1(\tau) I(t - \tau)$$

$$+ \int d\tau \int d\tau' F_2(\tau, \tau') I(t-\tau) I(t-\tau') + \cdots \qquad (2)$$

In the low SNR limit the rods respond linearly (they count photons), and we expect that the linear term dominates the series (2). We then solve analytically for the filter $F_1(\tau)$ which minimizes $\chi^2 = \langle \int dt\, |R(t) - R_{est}(t)|^2 \rangle$ — i.e. the filter which satisfies $\delta\chi^2/\delta F_1(\tau) = 0$. The averages $\langle \cdots \rangle$ are taken over the ensemble of stimuli

$R(t)$. The result of this optimization is[*]

$$F_1(\tau) = \int \frac{d\omega}{2\pi} e^{-i\omega\tau} \frac{\left\langle \tilde{R}(\omega)\tilde{I}^*(\omega) \right\rangle}{\left\langle |\tilde{I}(\omega)|^2 \right\rangle}.$$  (3)

In the photon counting regime the rod currents are described as a sum of impulse responses $I_0(t-t_\mu)$ occuring at the photon arrival times $t_\mu$, plus a noise term $\delta I(t)$. Expanding for low SNR we find

$$F_1(\tau) = \int \frac{d\omega}{2\pi} e^{-i\omega\tau} S_R(\omega) \frac{\tilde{I}_0^*(\omega)}{S_I(\omega)} + \cdots,$$  (4)

where $S_R(\omega)$ is the spectral density of fluctuations in the photon arrival rate, $\tilde{I}_0(\omega)$ is the Fourier transform of $I_0(t)$, and $S_I(\omega)$ is the spectral density of current noise $\delta I(t)$ in the rod.

The filter (4) naturally separates into two distinct stages: A "first" stage

$$\tilde{F}_{\text{bip}}(\omega) \equiv \tilde{I}_0^*(\omega)/S_{\text{I}}(\omega)$$  (5)

which depends only on the signal and noise properties of the rod cell, and a "second" stage $S_R(\omega)$ which contains our *a priori* knowledge of the stimulus. The first stage filter is the matched filter given the rod signal and noise characteristics; each frequency component in the output of this filter is weighted according to its input SNR.

Recall from the probabilistic argument above that optimal estimation of some arbitrary aspect of the scene, such as motion, also results in a separation into two processing stages. Specifically, estimation of *any* functional of light intensity involves only a filtered version of the rod currents. This filter is precisely the universal filter $F_{\text{bip}}(\tau)$ defined in (5). This result makes intuitive sense since the first stage of filtering is simply "cleaning up" the rod signals prior to subsequent computation. Intuitively we expect that this filtering occurs at an early stage of visual processing. The first opportunity to filter the rod signals occurs in the transfer of signals between the rod and bipolar cells; we identify the transfer function between these cells with the first stage of our optimal filter. More precisely we identify the intracellular voltage response of the bipolar cell with the output of the filter $F_{\text{bip}}(\tau)$. In response to a dim flash of light at $t = 0$ the average bipolar cell voltage response should then be

$$V_{\text{bip}}(t) \propto \int d\tau \, F_{\text{bip}}(\tau) I_0(t-\tau).$$  (6)

*Nowhere in this prediction process do we insert any information about the bipolar response — the shape of our prediction is governed entirely by signal and noise properties of the rod cell and the theoretical principle of optimality.*

## 3    Extracting the filter parameters and predicting the bipolar response

To complete our prediction of the dim flash bipolar response we extract the rod single photon current $I_0(t)$ and rod current noise spectrum $S_I(\omega)$ from experimen-

---

[*]We define the Fourier Transform as $\tilde{f}(\omega) = \int dt e^{+i\omega t} f(t)$.

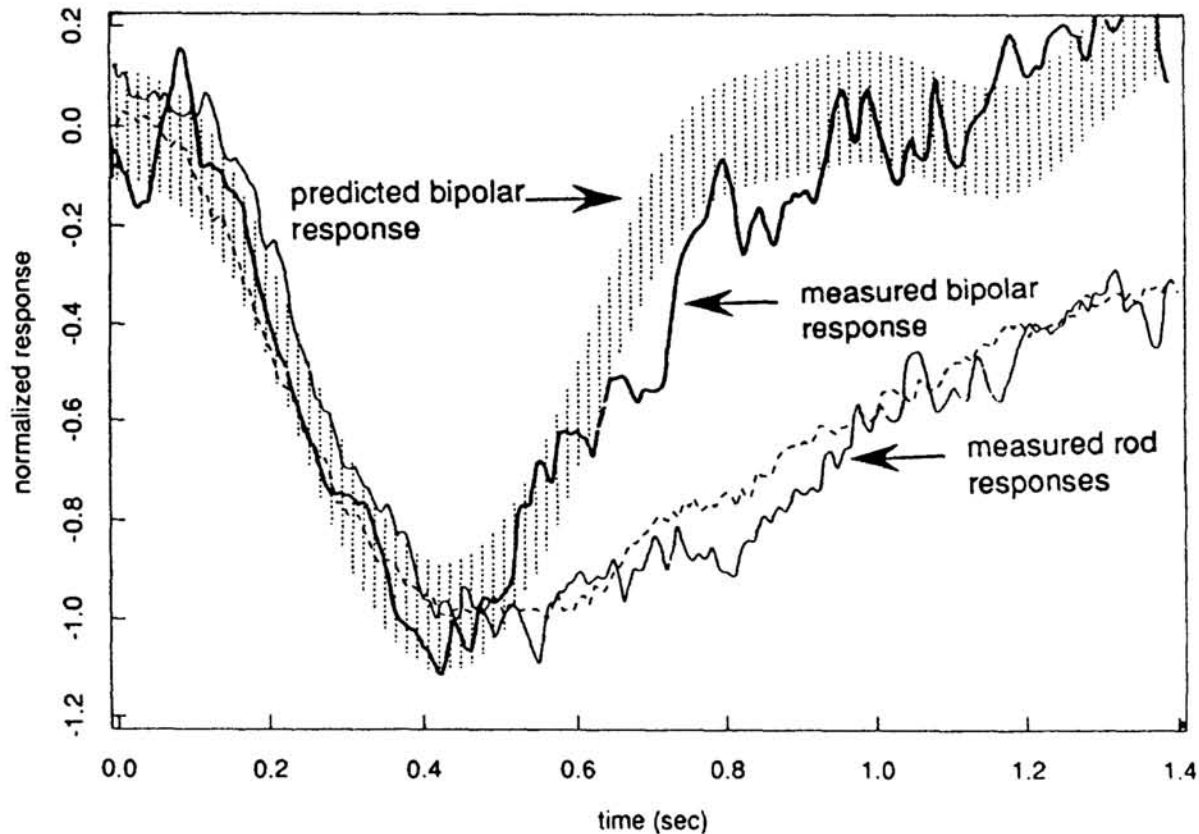

Figure 2: Comparison of predicted dim flash bipolar voltage response (based entirely on rod signal and noise characteristics) and measured bipolar voltage response. For reference we show rod voltage responses from two different cells which show the typical variations from cell to cell and thus indicate the variations we should expect in different bipolar cells. The measured responses are averages of many presentations of a diffuse flash occurring at $t = 0$ and resulting in the absorption of an average of about 5 photons in the rod cell. The errors bars are one standard deviation.

tal data. To compare our prediction directly with experiment we must obtain the rod characteristics under identical recording conditions as the bipolar measurement. This excludes suction pipette measurements which measure the currents directly, but effect the rod response dynamics [10,11]. The bipolar voltage response is measured intracellularly in the eyecup preparation [8]; our approach is to use intracellular voltage recordings to characterize the rod network and thus convert voltages to currents, as in [12]. This approach to the problem may seem overly complicated — why did we formulate the theory in terms of currents and not voltages? It is important we formulate our theory in terms of the *individual* rod signal and noise characteristics. The electrical coupling between rod cells in the retina causes the voltage noise in nearby rods to be correlated; each rod, however, independently injects current noise into the network.

The impedances connecting adjacent rod cells, the impedance of the rod cell itself and the spatial layout and connections between rods determine the relationship between currents and voltages in the network. The rods lie nearly on a square

lattice with lattice constant $20\,\mu$m. Using this result we extract the impedances from two independent experiments [12]. Once we have the impedances we "decorrelate" the voltage noise to calculate the uncorrelated current noise. We also convert the measured single photon voltage response to the corresponding current $I_0(t)$. It is important to realize that the impedance characteristics of the rod network are experimentally determined, and are not in any sense free parameters!

After completing these calculations the elements of our bipolar prediction are obtained under identical conditions to the experimental bipolar response, and we can make a direct comparison between the two; *there are no free parameters in this prediction.* As shown in the Fig. 2, the predicted bipolar response (6) is in excellent agreement with the measured response; all deviations are well within the error bars.

## 4   Concluding remarks

We began by posing a theoretical question: How can we best recover the photon arrival rate from observations of the rod signals? The answer, in the form of a linear filter which we apply to the rod current, divides into two stages — a stage which is matched to the rod signal and noise characteristics, and a stage which depends on the particular characteristics of the photon source we are observing. The first-stage filter in fact is the universal pre-processor for *all* visual processing tasks at low SNR. We identified this filter with the rod-bipolar transfer function, and based on this hypothesis predicted the bipolar response to a dim, diffuse flash. Our prediction agrees extremely well with experimental bipolar responses. We emphasize once more that this is not a "model" of the bipolar cell; in fact there is nothing in our theory about the physical properties of bipolar cells. Rather our approach results in parameter-free predictions of the *computational* properties of these cells from the general theoretical principle of optimal computation. As far as we know this is the first successful quantitative prediction from a theory of neural computation.

Thus far our results are limited to the dark-adapted regime; however the theoretical analysis presented here depends only on low SNR. This observation suggests a follow-up experiment to test the role of adaptation in the rod-bipolar transfer function. If the retina is first adapted to a constant background illumination and then shown dim flashes on top of the background we can use the analysis presented here to predict the *adapted* bipolar response from the *adapted* rod impulse response and noise. Such an experiments would answer a number of interesting questions about retinal processing: (1) Does the processing remain optimal at higher light levels? (2) Does the bipolar cell still function as the universal pre-processor? (3) Do the rod and bipolar cells adapt together in such a way that the optimal first-stage filter remains unchanged, or does the rod-bipolar transfer function also adapt?

Can these ideas be extended to other systems, particularly spiking cells? A number of other signal processing systems exhibit nearly optimal performance [2]. One example we are currently studying is the extraction of movement information from the array of photoreceptor voltages in the insect compound eye [13]. In related work, Atick and Redlich [14] have argued that the receptive field characteristics of retinal ganglion cells can be quantitatively predicted from a principle of optimal encoding (see also [15]). A more general question we are currently pursuing is the efficiency of the coding of sensory information in neural spike trains. Our

preliminary results indicate that the information rate in a spike train can be as high as 80% of the maximum information rate possible given the noise characteristics of spike generation [16]. From these examples we believe that "optimal performance" provides a general theoretical framework which can be used to predict the significant computational dynamics of cells in many neural systems.

## Acknowledgments

We thank R. Miller and W. Hare for sharing their data and ideas, D. Warland and R. de Ruyter van Steveninck for helping develop many of the methods we have used in this analysis, and J. Atick, J. Hopfield and D. Tank for many helpful discussions. W. B. thanks the Aspen Center for Physics for the environment which catalyzed these discussions. Work at Berkeley was supported by the National Institutes of Health through Grant No. EY 03785 to WGO, and by the National Science Foundation through a Presidential Young Investigator Award to WB, supplemented by funds from Cray Research, Sun Microsystems, and the NEC Research Institute, and through a Graduate Fellowship to FR.

## References

1. W. Bialek. *Ann. Rev. Biophys. Biophys. Chem.*, 16:455, 1987.

2. W. Bialek. In E. Jen, editor, *1989 Lectures in Complex Systems, SFI Studies in the Sciences of Complexity*, volume 2, pages 513–595. Addison-Wesley, Reading, Mass., 1990.

3. S. Hecht, S. Shlaer, and M. Pirenne. *J. Gen. Physiol.*, 25:819, 1942.

4. H. B. Barlow. *J. Opt. Soc. Am.*, 46:634, 1956.

5. H. B. Barlow. *Nature*, 334:296, 1988.

6. A.-C. Aho, K. Donner, C. Hydèn, L. O. Larsen, and T. Reuter. *Nature*, 324:348, 1988.

7. W. Bialek and W. Owen. *Biophys. J.*, in press.

8. W. A. Hare and W. G. Owen. *J. Physiol.*, 421:223, 1990.

9. M. Capovilla, W. A. Hare, and W. G. Owen. *J. Physiol.*, 391:125, 1987.

10. Denis Baylor, T. D. Lamb, and K.-W. Yau. *J. Physiol.*, 288:613–634, 1979.

11. D. Baylor, G. Matthews, and K. Yau. *J. Physiol.*, 309:591, 1980.

12. V. Torre and W. G. Owen. *Biophys. J.*, 41:305–324, 1983.

13. W. Bialek, F. Rieke, R. R. de Ruyter van Steveninck, and D. Warland. In D. Touretzky, editor, *Advances in Neural Information Processing Systems 2*, pages 36–43. Morgan Kaufmann, San Mateo, Ca., 1990.

14. J. J. Atick and N. Redlich. *Neural Computation*, 2:308, 1990.

15. W. Bialek, D. Ruderman, and A. Zee. In D. Touretzky, editor, *Advances in Neural Information Processing Systems 3*. Morgan Kaufmann, San Mateo, Ca., 1991.

16. F. Rieke, W. Yamada, K. Moortgat, E. R. Lewis, and W. Bialek. *Proceedings of the 9th International Symposium on Hearing*, 1991.
